# Localizing Bugs in Program Executions with Graphical Models

**Laura Dietz**
Max-Planck Institute for Computer Science
Saarbruecken, Germany
dietz@mpi-inf.mpg.de

**Valentin Dallmeier**
Saarland University
Saarbruecken, Germany
dallmeier@cs.uni-saarland.de

**Andreas Zeller**
Saarland University
Saarbruecken, Germany
zeller@cs.uni-saarland.de

**Tobias Scheffer**
Potsdam University
Potsdam, Germany
scheffer@cs.uni-potsdam.de

## Abstract

We devise a graphical model that supports the process of debugging software by guiding developers to code that is likely to contain defects. The model is trained using execution traces of passing test runs; it reflects the distribution over transitional patterns of code positions. Given a failing test case, the model determines the least likely transitional pattern in the execution trace. The model is designed such that Bayesian inference has a closed-form solution. We evaluate the *Bernoulli graph model* on data of the software projects AspectJ and Rhino.

## 1 Introduction

In today's software projects, two types of source code are developed: product and test code. Product code, also referred to as *the program*, contains all functionality and will be shipped to the customer. The program and its subroutines are supposed to behave according to a specification. The example program in Figure 1 (left), is supposed to always return the value 10. It contains a defect in line number 20, which lets it return a wrong value if the input variable equals five.

In addition to product code, developers write test code that consists of small test programs, each testing a single procedure or module for compliance with the specification. For instance, Figure 1 (right) shows three test cases, the second of which reveals the defect. Development environments provide support for running test cases automatically and would report failure of the second test case. Localizing defects in complex programs is a difficult problem because the failure of a test case confirms only the existence of a defect, not its location.

When a program is executed, its trace through the source code can be recorded. An executed line of source code is identified by a code position $s \in S$. The stream of code positions forms the trace $t$ of a test case execution. The data that our model analyses consists of a set $T$ of passing test cases $t$. In addition to the passing tests we are given a single trace $\bar{t}$ of a failing test case. The passing test traces and the trace of the failing case refer to the same code revision; hence, the semantics of each code position remain constant. For the failing test case, the developer is to be provided with a ranking of code positions according to their likelihood of being defective.

The semantics of code positions may change across revisions, and modifications of code may impact the distribution of execution patterns in the modified as well as other locations of the code. We focus on the problem of localizing defects within a current code revision. After each defect is localized, the code is typically revised and the semantics of code positions changes. Hence, in this setting, we

```
10 /**
11  * A procedure containing a defect.          public static class TestDefect extends TestCase {
12  *                                               public void testParam1() {
13  * @param param an arbitrary parameter.            assertEquals(10, defect(1));
14  * @return 10                                    }
15  */
16 public static int defect (int param) {          /** Failing test case. */
17     int i = 0;                                   public void testParam5() {
18     while (i < 10) {                                assertEquals(10, defect(5));
19        if (param == 5) {                         }
20            return 100;
21        }                                         public void testParam10() {
22        i++;                                         assertEquals(10, defect(10));
23     }                                            }
24     return i;                                }
25 }
```

Figure 1: Example with product code (left) and test code (right).

cannot assume that any negative training data—that is, previous failing test cases of the same code revision—are available. For that reason, discriminative models do not lend themselves to our task.

Instead of representing the results as a ranked list of positions, we envision a tight integration in development environments. For instance, on failure of a test case, the developer could navigate between predicted locations of the defect, starting with top ranked positions.

So far, Tarantula [1] is the standard reference model for localizing defects in execution traces. The authors propose an interface widget for test case results in which a pixel represents a code position. The hue value of the pixel is determined by the number of failing and passing traces that execute this position and correlates with the likelihood that $s$ is faulty [1]. Another approach [2] includes return values and flags for executed code blocks and builds on sensitivity and increase of failure probability. This approach was continued in project *Holmes* [3] to include information about executed control flow paths. Andrzejewski et al. [4] extend latent Dirichlet allocation (LDA) [5] to find bug patterns in recorded execution events. Their probabilistic model captures low-signal bug patterns by explaining passing executions from a set of usage topics and failing executions from a mix of usage and bug topics. Since a vast amount of data is to be processed, our approach is designed to not require estimating latent variables during prediction as is necessary with LDA-based approaches [4].

**Outline.** Section 2 presents the Bernoulli graph model, a graphical, generative model that explains program executions. This section's main result is the closed-form solution for Bayesian inference of the likelihood of a transitional pattern in a test trace given example execution traces. Furthermore, we discuss how to learn hyperparameters and smoothing coefficients from other revisions, despite the fragile semantics of code positions. In Section 3, reference methods and simpler probabilistic models are detailed. Section 4 reports on the prediction performance of the studied models for the AspectJ and Rhino development projects. Section 5 concludes.

## 2  Bernoulli Graph Model

The Bernoulli graph model is a probabilistic model that generates program execution graphs. In contrast to an execution trace, the graph is a representation of an execution that abstracts from the number of iterations over code fragments. The model allows for Bayesian inference of the likelihood of a transition between code positions within an execution, given previously seen executions.

The $n$-gram execution graph $G_t = (V_t, E_t, L_t)$ of an execution $t$ connects vertices $V_t$ by edges $E_t \subseteq V_t \times V_t$. Labeling function $L_t : V_t \to S^{(n-1)}$ injectively maps vertices to $n-1$-grams of code positions, where $S$ is the alphabet of code positions.

In the bigram execution graph, each vertex $v$ represents a code position $L_t(v)$; each arc $(u, v)$ indicates that code position $L_t(v)$ has been executed directly after code position $L_t(u)$ at least once during the program execution. In $n$-gram execution graphs, each vertex $v$ represents a fragment $L_t(v) = s_1 \ldots s_{n-1}$ of consecutively executed statements. Vertices $u$ and $v$ can only be connected by an arc if the fragments are overlapping in all but the first code position of $u$ and the last code position of $v$; that is, $L_t(u) = s_1 \ldots s_{n-1}$ and $L_t(v) = s_2 \ldots s_n$. Such vertices $u$ and $v$ are

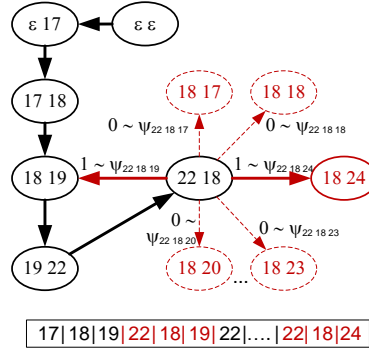

Figure 2: Expanding vertex "22 18" in the generation of a tri-gram execution graph corresponding to the trace at the bottom. Graph before expansion is drawn in black, new parts are drawn in solid red.

connected by an arc if code positions $s_1 \ldots s_n$ are executed consecutively at least once during the execution. For the example program in Figure 1 the tri-gram execution graph is given in Figure 2.

**Generative process.** The Bernoulli graph model generates one graph $G_{m,t} = (V_{m,t}, E_{m,t}, L_{m,t})$ per execution $t$ and procedure $m$. The model starts the graph generation with an initial vertex representing a fragment of virtual code positions $\varepsilon$.

In each step, it expands a vertex $u$ labeled $L_{m,t}(u) = s_1 \ldots s_{n-1}$ that has not yet been expanded; *e.g.*, vertex "22 18" in Figure 2. Expansion proceeds by tossing a coin with parameter $\psi_{m,s_1\ldots s_n}$ for each appended code position $s_n \in S$. If the coin toss outcome is positive, an edge to vertex $v$ labeled $L_{m,t}(v) = s_2 \ldots s_n$ is introduced. If $V_{m,t}$ does not yet include a vertex $v$ with this labeling, it is added at this point. Each vertex is expanded only once. The process terminates if no vertex is left that has been introduced but not yet expanded. Parameters $\psi_{m,s_1\ldots s_n}$ are governed by a Beta distribution with fixed hyperparameters $\alpha_\psi$ and $\beta_\psi$. In the following we focus on the generation of edges, treating the vertices as observed. Figure 3a) shows a factor graph representation of the generative process and Algorithm 1 defines the generative process in detail.

**Inference.** Given a collection $\mathcal{G}_m$ of previously seen execution graphs for method $m$ and a new execution $G_m = (V_m, E_m, L_m)$, Bayesian inference determines the likelihood $p((u, v) \in E_m | V_m, \mathcal{G}_m, \alpha_\psi, \beta_\psi)$ of each of the edges $(u, v)$, thus indicating unlikely transitions in the new execution of $m$ represented by execution graph $G_m$. Since we employ independent models for all

---

**Algorithm 1** Generative process of the Bernoulli graph model.

  **for all** procedures $m$ **do**
    **for all** $s_1...s_n \in (S_m)^n$ **do**
      **draw** $\psi_{m,s_1...s_n} \sim Beta(\alpha_\psi, \beta_\psi)$.
    **for all** executions $t$ **do**
      create a new graph $G_{m,t}$.
      add a vertex $u$ labeled $\varepsilon\varepsilon...\varepsilon$.
      initialize queue $Q = \{u\}$.
      **while** queue $Q$ is not empty **do**
        dequeue $u \leftarrow Q$, with $L(u) = s_1 \ldots s_{n-1}$.
        **for all** $s_n \in S_m$ **do**
          let $v$ be a vertex with $L(v) = s_2 \ldots s_n$.
          **draw** $b \sim Bernoulli(\psi_{m,s_1...s_n})$.
          **if** $b = 1$ **then**
            **if** $v \notin V_{m,t}$ **then**
              add $v$ to $V_{m,t}$.
              enqueue $v \rightarrow Q$.
            add arc $(u, v)$ to $E_{m,t}$.

---

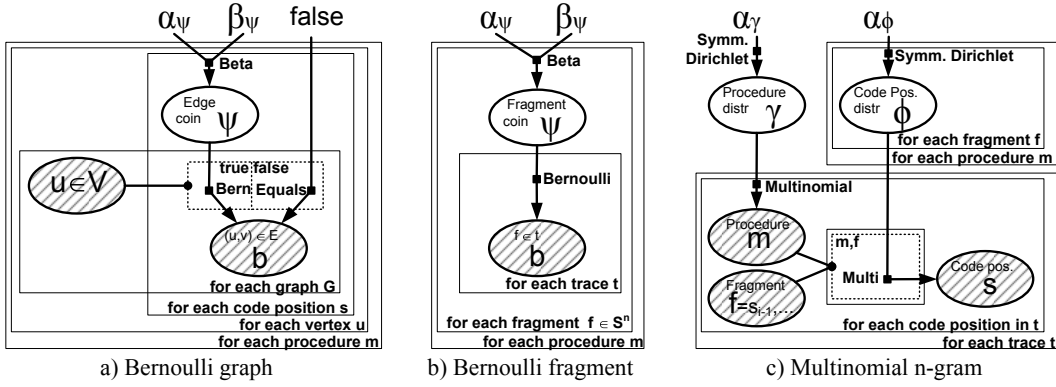

a) Bernoulli graph      b) Bernoulli fragment      c) Multinomial n-gram

Figure 3: Generative models in directed factor graph notation with dashed rectangles indicating gates [6].

methods $m$, inference can be carried out for each method separately. Since vertices $V_m$ are observed, coin parameters $\Psi$ are d-separated from each other (cf. Figure 3a). We yield independent Beta-Bernoulli models conditioned on the presence of start vertices $u$. Thus, predictive distributions for presence of edges in future graphs can be derived in closed form (Equation 1) where $\#_u^{\mathcal{G}}$ denotes the number of training graphs containing vertices labeled $L(u)$ and $\#_{(u,v)}^{\mathcal{G}}$ denotes the number of training graphs containing edges between vertices labeled $L(u)$ and $L(v)$. See the appendix for a detailed derivation of Equation 1.

$$p((u,v) \in E_m | V_m, \mathcal{G}_m, \alpha_\psi, \beta_\psi) = \frac{\#_{(u,v)}^{\mathcal{G}} + \alpha_\psi}{\#_u^{\mathcal{G}} + \alpha_\psi + \beta_\psi}. \tag{1}$$

By definition, an execution graph $G$ for an execution contains a vertex if its label is a substring of the execution's trace $t$. Likewise, an edge is contained if an aggregation of the vertex labels is a substring of $t$. It follows[1] that the predictive distribution can be reformulated as in Equation 2 to predict the probability of seeing the code position $\tilde{s} = s_n$ after a fragment of preceding statements $\tilde{f} = s_1 \ldots s_{n-1}$ using the trace representation of an execution. Thus, it is not neccessary to represent execution graphs $G$ explicitly.

$$p(\tilde{s}|\tilde{f}, T, \alpha_\psi, \beta_\psi) = \frac{\#\{t \in T | \tilde{f}\tilde{s} \in t\} + \alpha_\psi}{\#\{t \in T | \tilde{f} \in t\} + \alpha_\psi + \beta_\psi} \tag{2}$$

**Estimating interpolation coefficients and hyperparameters.** For given hyperparameters and fixed context length $n$, Equation 2 predicts the likelihood for $\tilde{s}_i$ following a fragment $\tilde{f} = \tilde{s}_{i-1} \ldots \tilde{s}_{i-n+1}$. To avoid sparsity issues while maintaining good expressiveness, we smooth various context lengths up to $N$ by interpolation.

$$p(\tilde{s}_i | \tilde{s}_{i-1} \ldots \tilde{s}_{i-N+1}, T, \alpha_\psi, \beta_\psi, \theta) = \sum_{n=1}^{N} p(n|\theta) \cdot p(\tilde{s}_i | \tilde{s}_{i-1} \ldots \tilde{s}_{i-n+1}, T, \alpha_\psi, \beta_\psi)$$

We can learn from different revisions by integrating multiple Bernoulli graphs models in a generative process, in which coin parameters are not shared across revisions and context lengths $n$. This process generates a stream of statements with defect flags. We learn hyperparameters $\alpha_\psi$ and $\beta_\psi$ jointly with $\theta$ using an automatically derived Gibbs sampling algorithm [7].

**Predicting defective code positions.** Having learned point estimates for $\hat{\alpha}_\psi$, $\hat{\beta}_\psi$, and $\hat{\theta}$ from other revisions in a leave-one-out fashion, statements $\tilde{s}$ are scored by the complementary event of being normal for any preceding fragment $\tilde{f}$.

$$score(\tilde{s}) = \max_{\tilde{f} \text{ preceding } \tilde{s}} \left( 1 - p(\tilde{s}|\tilde{f}, T, \hat{\alpha}_\psi, \hat{\beta}_\psi, \hat{\theta}) \right) \tag{3}$$

The maximum is justified because an erroneous code line may show its defective behavior only in combination with some preceding code fragments, and even a single erroneous combination is enough to lead to defective behavior of the software.

# 3 Reference Methods

The Tarantula model is a popular scoring heuristic for defect localization in software engineering. We will prove a connection between Tarantula and the unigram variant of a Bernoulli graph model. Furthermore, we will discuss other reference models which we will consider in the experiments.

## 3.1 Tarantula

Tarantula [1] scores the likelihood of a code position $s$ being defective according to the proportions of failing $F$ and passing traces $T$ that execute this position (Equation 4).

$$score^{Tarantula}(\tilde{s}) = \frac{\frac{\#\{\bar{t} \in F | \tilde{s} \in \bar{t}\}}{\#\{\bar{t} \in F\}}}{\frac{\#\{\bar{t} \in F | \tilde{s} \in \bar{t}\}}{\#\{\bar{t} \in F\}} + \frac{\#\{t \in T | \tilde{s} \in t\}}{\#\{t \in T\}}} \tag{4}$$

For the case that only one test case fails, we can show an interesting relationship between Tarantula, the unigram Bernoulli graph model, and multivariate Bernoulli models (referred to in [8]). In the unigram case, the Bernoulli graph model generates a graph in which all statements in an execution are directly linked to an empty start vertex. In this case, the Bernoulli graph model is equal to a multi-variate Bernoulli model generating a set of statements for each execution.

Using an improper prior $\alpha_\psi = \beta_\psi = 0$, the unigram Bernoulli graph model scores a statement by $score^{Graph}(\tilde{s}) = 1 - \frac{\#\{t \in T | \tilde{s} \in t\}}{\#\{t \in T\}}$. Letting $g(s) = \frac{\#\{t \in T | \tilde{s} \in t\}}{\#\{t \in T\}}$, the rank order of any two code positions $s_1$, $s_2$ is determined by $1 - g(s_1) > 1 - g(s_2)$ or equivalently $\frac{1}{1+g(s_1)} > \frac{1}{1+g(s_2)}$ which is Tarantula's ranking criterion if $\#F$ is 1.

## 3.2 Bernoulli Fragment Model

Inspired by this equivalence, we study a naive $n$-gram extension to multi-variate Bernoulli models which we call Bernoulli fragment model. Instead of generating a set of statements, the Bernoulli model may generate a set of fragments for each execution.

Given a fixed order $n$, the Bernoulli fragment model draws a coin parameter for each possible fragment $f = s_1 \ldots s_n$ over the alphabet $S_m$. For each execution the fragment set is generated by tossing a fragment's coin and including all fragments with outcome $b = 1$ (cf. Figure 3b). The probability of an unseen fragment $\tilde{f}$ is given by $p(\tilde{f}|T, \alpha_\psi, \beta_\psi) = \frac{\#\{t \in T | \tilde{f} \in t\} + \alpha_\psi}{\#\{t \in T\} + \alpha_\psi + \beta_\psi}$.

The model deviates from reality in that it may generate fragments that may not be aggregateable into a consistent sequence of code positions. Thus, non-zero probability mass is given to impossible events, which is a potential source of inaccuracy.

## 3.3 Multinomial Models

The multinomial model is popular in the text domain—*e.g.,* [8]. In contrast to the Bernoulli graph model, the multinomial model takes the number of occurrences of a pattern within an execution into account. It consists of a hierarchical process in which first a procedure $m$ is drawn from multinomial distribution $\gamma$, then a code position $s$ is drawn from the multinomial distribution $\phi_m$ ranging over all code positions $S_m$ in the procedure.

The $n$-gram model is a well-known extension of the unigram multinomial model, where the distributions $\phi$ are conditioned on the preceding fragment of code positions $f = s_1 \ldots s_{n-1}$ to draw a follow-up statement $s_n \sim \phi_{m,f}$. Using fixed symmetric Dirichlet distributions with parameter $\alpha_\gamma$ and $\alpha_\phi$ as priors for the multinomial distributions, the probability for unseen code positions $\tilde{s}$ following on fragment $\tilde{f}$ is given in Equation 5. Shorthand $\#^T_{s \in m}$ denotes how often statements in procedure $m$ are executed (summing over all traces $t \in T$ in the training set); and $\#^T_{m,s_1 \ldots s_n}$ denotes the number times statements $s_1 \ldots s_n$ are executed subsequently by procedure $m$.

$$p(\tilde{s}, \tilde{m}|\tilde{f}, T, \alpha_\gamma, \alpha_\phi) \propto \underbrace{\frac{\#^T_{s \in \tilde{m}} + \alpha_\gamma}{\sum_{m' \in M} \#^T_{s \in m'} + \alpha_\gamma \#M}}_{\gamma(\tilde{m})} \cdot \underbrace{\frac{\#^T_{\tilde{m}, \tilde{f}\tilde{s}} + \alpha_\phi}{\#^T_{\tilde{m}, \tilde{f}} + \alpha_\phi \#S_{\tilde{m}}}}_{\phi_{\tilde{m}, \tilde{f}}(\tilde{s})} \tag{5}$$

### 3.4 Holmes

Chilimbi et al. [3] propose an approach that relies on a stream of sampled boolean predicates $P$, each corresponding to an executed control flow branch starting at code position $s$. The approach evaluates whether $P$ being true increases the probability of failure in contrast to reaching the code position by chance. Each code position is scored according to the importance of its predicate $P$ which is the harmonic mean of sensitivity and increase in failure probability. Shorthands $F_e(P)$ and $S_e(P)$ refer to the failing/passing traces that executed the path $P$, where $F_o(P)$ and $S_o(P)$ refer to failing/passing traces that executed the start point of $P$.

$$Importance(P) = \frac{2}{\frac{\log \#F}{\log F_e(P)} + \left( \frac{F_e(P)}{S_e(P)+F_e(P)} - \frac{F_o(P)}{S_o(P)+F_o(P)} \right)^{-1}}$$

This scoring procedure is not applicable to cases where a path is executed in only one failing trace, as a division by zero occurs in the first term when $F_e(P) = 1$. This issue renders Holmes inapplicable to our case study where typically only one test case fails.

### 3.5 Delta LDA

Andrzejewski et al. [4] use a variant of latent Dirichlet Allocation (LDA) [5] to identify topics of co-occurring statements. Most topics may be used to explain passing and failing traces, where some topics are reserved to explain statements in the failing traces only. This is obtained by running LDA with different Dirichlet priors on passing and failing traces. After inference, the topic specific statement distributions $\phi = p(s|z)$ are converted to $p(z|s)$ via Bayes' rule. Then statements $j$ are ranked according to the confidence $S_{ij} = p(z = i|s = j) - \max_{k \neq i} p(z = k|s = j)$ of being rather about a bug topic $i$ than any other topic $k$.

## 4 Experimental Evaluation

In this section we study empirically how accurately the Bernoulli graph model and the reference models discussed in Section 3 localize defects that occurred in two large-scale development projects.

We find that data used for previous studies is not appropriate for our investigation. The SIR repository [9] provides traces of small programs into which defects have been injected. However, as pointed out in [10], there is no strong argument as to why results obtained on specifically designed programs with artificial defects should necessarily transfer to realistic software development projects with actual defects. The Cooperative Bug Isolation project [11], on the other hand, collects execution data from real applications, but records only a random sample of 1% of the executed code positions; complete execution traces cannot be reconstructed. Therefore, we use the development history of two large-scale open source development projects, AspectJ and Rhino, as gathered in [12].

**Data set.**  From Rhino's and AspectJ's bug database, we select defects which are reproducable by a test case and identify corresponding revisions in the source code repository. For such revisions, the test code contains a test case that fails in one revision, but passes in the following revision. We use the code positions that were modified between the two revisions as ground truth for the defective code positions $\mathcal{D}$. For AspectJ, these are one or two lines of code; the Rhino project contains larger code changes. For each such revision, traces $T$ of passing test cases are recorded on a line number basis. In the same manner, the failing trace $\bar{t}$ (in which the defective code is to be identified) is recorded.

The AspectJ data set consists of 41 defective revisions and a total of 45 failing traces. Each failing trace has a length of up to 2,000,000 executed statements covering approx. 10,000 different code positions (of the 75,000 lines in the project), spread across 300 to 600 files and 1,000 to 4,000 procedures. For each revision, we recorded 100 randomly selected valid test cases (drawn out of approx. 1000).

Rhino consists of 15 defective revisions with one failing trace per bug. Failing traces have an average length of 3,500,000 executed statements, covering approx. 2,000 of 38,000

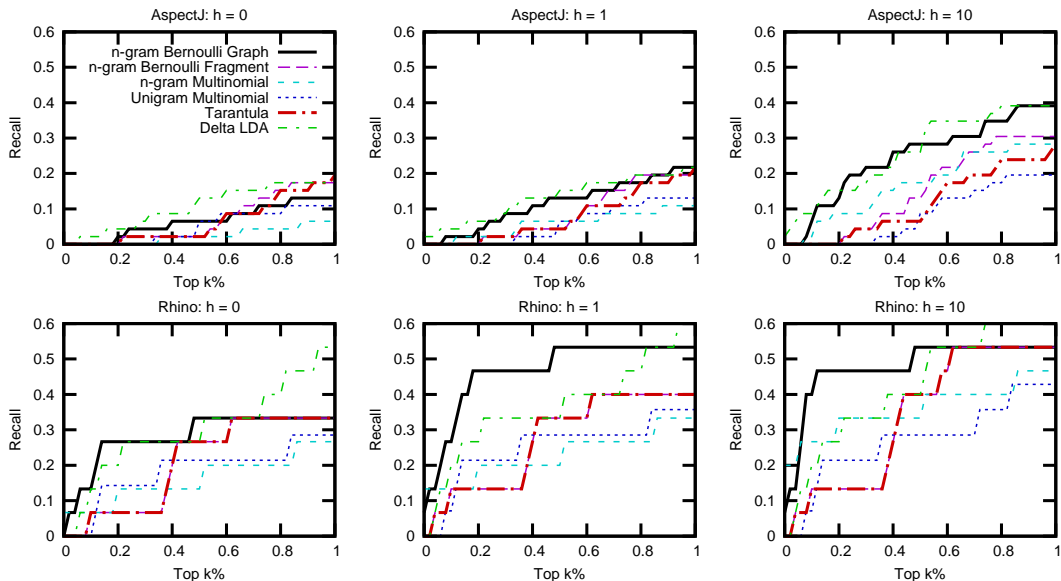

Figure 4: Recall of defective code positions within the 1% highest scored statements for AspectJ (top) and Rhino (bottom), for windows of $h = 0$, $h = 1$, and $h = 10$ code lines.

code positions, spread across 70 files and 650 procedures. We randomly selected 100 of the 1500 valid traces for each revision as training data. Both data sets are available at http://www.mpi-inf.mpg.de/~dietz/debugging.html.

**Evaluation criterion.** Following the evaluation in [1], we evaluate how well the models are able to guide the user into the vicinity of a defective code position. The models return a ranked list of code positions. Envisioning that the developer can navigate from the ranking into the source code to inspect a code line within its context, we evaluate the rank $k$ at which a line of code occurs that lies within a window of $\pm h$ lines of code of a defective line. We plot relative ranks; that is, absolute ranks divided by the number of covered code lines, corresponding to the fraction of code that the developer has to walk through in order to find the defect. We examine the recall@$k$%, that is the fraction of successfully localized defects over the fraction of code the user has to inspect. We expect a typical developer to inspect the top 0.25% of the ranking, corresponding to approximately 25 ranks for AspectJ.

Neither the AUC nor the Normalized Discounted Cummulative Gain (NDCG) appropriately measure performance in our application. AUC does not allow for a cut-off rank; NDCG will inappropriately reward cases in which many statements in a defect's vicinity are ranked highly.

**Reference methods.** In order to study the helpfulness of each generative model, we evaluate smoothed models with maximum length $N = 5$ for each the multinomial, Bernoulli fragment and Bernoulli graph model. We compare those to the unigram multinomial model and Tarantula. Tuning and prediction of reference methods follow in accordance to Section 2. In addition, we compare to the latent variable model Delta LDA with nine usage and one bug topics, $\alpha = 0.5$, $\beta = 0.1$, and 50 sampling iterations.

**Results.** The results are presented in Figure 4. The Bernoulli graph model is always ahead of reference methods that have a closed form solution in the top 0.25% and top 0.5% of the ranking. This improvement is significant with level 0.05 in comparison to Tarantula for $h = 1$ and $h = 10$. It is significantly better than the $n$-gram multinomial model for $h = 1$. Although increasing $h$ makes the prediction problem generally easier, only Bernoulli graph and the multinomial $n$-gram model play to their strength.

A comparison by Area under the Curve in top 0.25% and top 0.5% indicates that the Bernoulli graph is more than twice as effective as Tarantula for the data sets for $h = 1$ and $h = 10$. Using the

Bernoulli graph model, a developer finds nearly every second bug in the top 1% in both data sets, where ranking a failing trace takes between 10 and 20 seconds.

According to a pair-t-test with 0.05-level, Bernoulli graph's prediction performance is significantly better than Delta LDA for the Rhino data set. No significant diffference is found for the AspectJ data set, but Delta LDA takes much longer to compute (approx. one hour versus 20 seconds) since parameters can not be obtained in closed form but require iterative sampling.

**Analysis.** Most revisions in our data sets had bugs that were equally difficult for most of the models. From revisions where one model drastically outperformed the others we identified different categories of suspicious code areas. In some cases, the defective procedures were executed in very few or no passing trace; we refer such code as being insufficiently covered. Another category refers to defective code lines in the vicinity of branching points such as if-statements. If code before the branch point is executed in many passing traces, but code in one of the branches only rarely, we call this a suspicious branch point.

The Bernoulli fragment model treats both kinds of suspicious code areas in a similar way. They have a different effect on the predictive Beta-posteriors in the Bernoulli graph model: insufficient coverage decreases the confidence; suspicious branch points will decrease the mean. The Beta-priors $\alpha_\psi$ and $\beta_\psi$ play a crucial role in weighting these two types of potential bugs in the ranking and encode prior beliefs on expecting one or the other. Our hyperparameter estimation procedure usually selects $\alpha_\psi = 1.25$ and $\beta_\psi = 1.03$ for all context lengths.

Revisions in which Bernoulli fragment outperformed Bernoulli graph contained defects in insufficiently covered areas. Presumably, Bernoulli graph identified many suspicious branching points, and assigned them a higher score. Revisions in which Bernoulli graph outperformed Bernoulli fragment contained bugs at suspicious branching points.

In contrast to the Bernoulli-style models, the multinomial models take the number of occurrences of a code position within a trace into account. Presumably, multiple occurrences of code lines within a trace do not indicate their defectiveness.

## 5   Conclusions

We introduced the Bernoulli graph model, a generative model that implements a distribution over program executions. The Bernoulli graph model generates $n$-gram execution graphs. Compared to execution traces, execution graphs abstract from the number of iterations that sequences of code positions have been executed for. The model allows for Bayesian inference of the likelihood of transitional patterns in a new trace, given execution traces of passing test cases. We evaluated the model and several less complex reference methods with respect to their ability to localize defects that occurred in the development history of AspectJ and Rhino. Our evaluation does not rely on artificially injected defects.

We find that the Bernoulli graph model outperforms Delta LDA on Rhino and performs as good as Delta LDA on the AspectJ project, but in substantially less time. Delta LDA is based on a multinomial unigram model, which performs worst in our study. This gives raise to the conjecture that Delta LDA might benefit from replacing the multinomial model with a Bernoulli graph model. this conjecture would need to be studied empirically.

The Bernoulli graph model outperforms the reference models with closed-form solution with respect to giving a high rank to code positions that lie in close vicinity of the actual defect. In order to find every second defect in the release history of Rhino, the Bernoulli graph model walks the developer through approximately 0.5% of the code positions and 1% in the AspectJ project.

**Acknowledgements**

Laura Dietz is supported by a scholarship of Microsoft Research Cambridge. Andreas Zeller and Tobias Scheffer are supported by a Jazz Faculty Grant.

## Footnotes

[1] For a set $A$ we denote its cardinality by $\#A$ rather than $|A|$ to avoid confusion with conditioned signs.

# References

[1] James A. Jones and Mary J. Harrold. Empirical evaluation of the tarantula automatic fault-localization technique. In *Proceedings of the International Conference on Automated Software Engineering*, 2005.

[2] Ben Liblit, Mayur Naik, Alice X. Zheng, Alex Aiken, and Michael I. Jordan. Scalable statistical bug isolation. In *Proceedings of the Conference on Programming Language Design and Implementation*, 2005.

[3] Trishul Chilimbi, Ben Liblit, Krishna Mehra, Aditya Nori, and Kapil Vaswani. Holmes: Effective statistical debugging via efficient path profiling. In *Proceedings of the International Conference on Software Engineering*, 2009.

[4] David Andrzejewski, Anne Mulhern, Ben Liblit, and Xiaojin Zhu. Statistical debugging using latent topic models. In *Proceedings of the European Conference on Machine Learning*, 2007.

[5] David M. Blei, Andrew Y. Ng, and Michael I. Jordan. Latent Dirichlet allocation. *Journal of Machine Learning Research*, 3:993–1022, 2003.

[6] Tom Minka and John Winn. Gates. In *Advances in Neural Information Processing Systems*, 2008.

[7] Hal Daume III. Hbc: Hierarchical Bayes Compiler. http://hal3.name/HBC, 2007.

[8] Andrew McCallum and Kamal Nigam. A comparison of event models for Naive Bayes text classification. In *Proceedings of the AAAI Workshop on Learning for Text Categorization*, 1998.

[9] Hyunsook Do, Sebastian Elbaum, and Gregg Rothermel. Supporting controlled experimentation with testing techniques: An infrastructure and its potential impact. *Empirical Software Engineering*, 10(4):405–435, October 2005.

[10] Lionel C. Briand. A critical analysis of empirical research in software testing. In *Proceedings of the Symposium on Empirical Software Engineering and Measurement*, 2007.

[11] Ben Liblit, Mayur Naik, Alice X. Zheng, Alex Aiken, and Michael I. Jordan. Public deployment of cooperative bug isolation. In *Proceedings of the Workshop on Remote Analysis and Measurement of Software Systems*, 2004.

[12] Valentin Dallmeier and Thomas Zimmermann. Extraction of bug localization benchmarks from history. In *Proceedings of the International Conference on Automated Software Engineering*, 2007.

